# Perfect Dimensionality Recovery
# by Variational Bayesian PCA

**Shinichi Nakajima**
Nikon Corporation
Tokyo, 140-8601, Japan
nakajima.s@nikon.co.jp

**Ryota Tomioka**
The University of Tokyo
Tokyo 113-8685, Japan
tomioka@mist.i.u-tokyo.ac.jp

**Masashi Sugiyama**
Tokyo Institute of Technology
Tokyo 152-8552, Japan
sugi@cs.titech.ac.jp

**S. Derin Babacan**
University of Illinois at Urbana-Champaign
Urbana, IL 61801, USA
dbabacan@illinois.edu

## Abstract

The variational Bayesian (VB) approach is one of the best tractable approximations to the Bayesian estimation, and it was demonstrated to perform well in many applications. However, its good performance was not fully understood theoretically. For example, VB sometimes produces a sparse solution, which is regarded as a practical advantage of VB, but such sparsity is hardly observed in the rigorous Bayesian estimation. In this paper, we focus on probabilistic PCA and give more theoretical insight into the empirical success of VB. More specifically, for the situation where the noise variance is unknown, we derive a sufficient condition for perfect recovery of the true PCA dimensionality in the large-scale limit when the size of an observed matrix goes to infinity. In our analysis, we obtain bounds for a noise variance estimator and simple closed-form solutions for other parameters, which themselves are actually very useful for better implementation of VB-PCA.

## 1 Introduction

Variational Bayesian (VB) approximation [1] was proposed as a computationally efficient alternative to rigorous Bayesian estimation. The key idea is to force the posterior to be factorized, so that the integration—a typical intractable operation in Bayesian methods—can be analytically performed over each parameter with the other parameters fixed. VB has been successfully applied to many applications [4, 7, 20, 11].

Typically, VB solves a non-convex optimization problem with an iterative algorithm [3], which makes theoretical analysis difficult. An important exceptional case is the matrix factorization (MF) model [11, 6, 19] with no missing entry in the observed matrix. Recently, the global analytic solution of VBMF has been derived and theoretical properties such as the mechanism of sparsity induction have been revealed [15, 16]. These works also posed thought-provoking relations between VB and rigorous Bayesian estimation: The VB posterior is actually quite different from the true Bayes posterior (compare the left and the middle graphs in Fig. 1), and VB induces sparsity in its solution but such sparsity is hardly observed in rigorous Bayesian estimation (see the right graph in Fig. 1).[1] These facts might deprive the justification of VB based solely on the fact that it is one of the best tractable approximations to the Bayesian estimation.

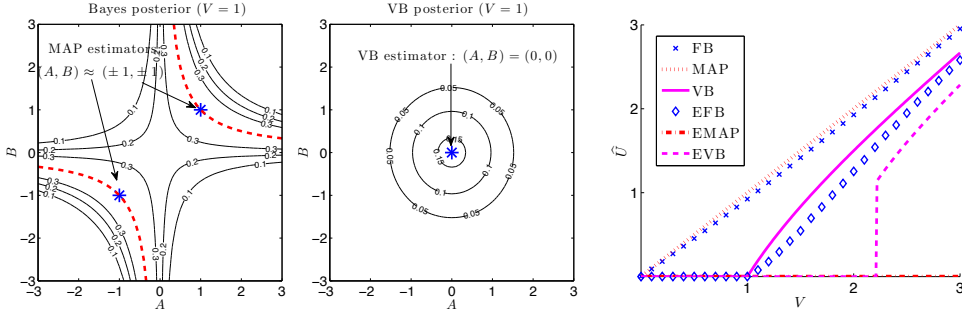

Figure 1: Dissimilarities between VB and the rigorous Bayesian estimation. (Left and Center) the Bayes posterior and the VB posterior of a $1 \times 1$ MF model, $V = BA + \mathcal{E}$, when $V = 1$ is observed ($\mathcal{E}$ is a Gaussian noise). VB approximates the Bayes posterior having two modes by an origin-centered Gaussian, which induces sparsity. (Right) Behavior of estimators of $U = BA^\top$, given the observation $V$. The VB estimator (the magenta solid curve) is zero when $V \leq 1$, which means *exact* sparsity. On the other hand, FB (fully-Bayesian or rigorous Bayes; blue crosses) shows no sign of sparsity. Further discussion will be provided in Section 5.2. All graphs are quoted from [15].

Since the probabilistic PCA [21, 18, 3] is an instance of MF, the global analytic solution derived in [16] for MF can be utilized for analyzing the probabilistic PCA. Indeed, automatic dimensionality selection of VB-PCA, which is an important practical advantage of VB-PCA, was theoretically investigated in [17]. However, the noise variance, which is usually unknown in many realistic applications of PCA, was treated as a given constant in that analysis.[2] In this paper, we consider a more practical and challenging situation where the noise variance is unknown, and theoretically analyze VB-PCA.

It was reported that noise variance estimation fails in some Bayesian approximation methods, if the formal rank is set to be full [17]. With such methods, an additional model selection procedure is required for dimensionality selection [14, 5]. On the other hand, we theoretically show in this paper that VB-PCA can estimate the noise variance accurately, and therefore automatic dimensionality selection works well. More specifically, we establish a sufficient condition that VB-PCA can *perfectly* recover the true dimensionality in the *large-scale limit* when the size of an observed matrix goes to infinity. An interesting finding is that, although the objective function minimized for noise variance estimation is multimodal in general, only a local search algorithm is required for perfect recovery. Our results are based on the random matrix theory [2, 5, 13, 22], which elucidates the distribution of singular values in the large-scale limit.

In the development of the above theoretical analysis, we obtain bounds for the noise variance estimator and simple closed-form solutions for other parameters. We also show that they can be nicely utilized for better implementation of VB-PCA.

## 2 Formulation

In this section, we introduce the variational Bayesian matrix factorization (VBMF).

### 2.1 Bayesian Matrix Factorization

Assume that we have an observation matrix $V \in \mathbb{R}^{L \times M}$, which is the sum of a target matrix $U \in \mathbb{R}^{L \times M}$ and a noise matrix $\mathcal{E} \in \mathbb{R}^{L \times M}$:

$$V = U + \mathcal{E}.$$

In the *matrix factorization* model, the target matrix is assumed to be low rank, which can be expressed as the following factorizability:

$$U = BA^\top,$$

where $A \in \mathbb{R}^{M \times H}$ and $B \in \mathbb{R}^{L \times H}$. $\top$ denotes the transpose of a matrix or vector. Thus, the rank of $U$ is upper-bounded by $H \leq \min(L, M)$.

In this paper, we consider the *probabilistic matrix factorization* (MF) model [19], where the observation noise $\mathcal{E}$ and the priors of $A$ and $B$ are assumed to be Gaussian:

$$p(V|A, B) \propto \exp\left(-\tfrac{1}{2\sigma^2}\|V - BA^\top\|_{\text{Fro}}^2\right), \tag{1}$$

$$p(A) \propto \exp\left(-\tfrac{1}{2}\text{tr}\left(AC_A^{-1}A^\top\right)\right), \qquad p(B) \propto \exp\left(-\tfrac{1}{2}\text{tr}\left(BC_B^{-1}B^\top\right)\right). \tag{2}$$

Here, we denote by $\|\cdot\|_{\text{Fro}}$ the Frobenius norm, and by $\text{tr}(\cdot)$ the trace of a matrix. We assume that $L \leq M$. If $L > M$, we may simply re-define the transpose $V^\top$ as $V$ so that $L \leq M$ holds.[3] Thus, this does not impose any restriction. We assume that the prior covariance matrices $C_A$ and $C_B$ are diagonal and positive definite, i.e.,

$$C_A = \text{diag}(c_{a_1}^2, \ldots, c_{a_H}^2), \qquad C_B = \text{diag}(c_{b_1}^2, \ldots, c_{b_H}^2)$$

for $c_{a_h}, c_{b_h} > 0, h = 1, \ldots, H$. Without loss of generality, we assume that the diagonal entries of the product $C_A C_B$ are arranged in non-increasing order, i.e., $c_{a_h} c_{b_h} \geq c_{a_{h'}} c_{b_{h'}}$ for any pair $h < h'$.

Throughout the paper, we denote a column vector of a matrix by a bold lowercase letter, and a row vector by a bold lowercase letter with a tilde, namely,

$$A = (\boldsymbol{a}_1, \ldots, \boldsymbol{a}_H) = (\widetilde{\boldsymbol{a}}_1, \ldots, \widetilde{\boldsymbol{a}}_M)^\top \in \mathbb{R}^{M \times H}, B = (\boldsymbol{b}_1, \ldots, \boldsymbol{b}_H) = \left(\widetilde{\boldsymbol{b}}_1, \ldots, \widetilde{\boldsymbol{b}}_L\right)^\top \in \mathbb{R}^{L \times H}.$$

## 2.2 Variational Bayesian Approximation

The Bayes posterior is given by

$$p(A, B|V) = \tfrac{p(V|A,B)p(A)p(B)}{p(V)}, \tag{3}$$

where $p(Y) = \langle p(V|A, B) \rangle_{p(A)p(B)}$. Here, $\langle \cdot \rangle_p$ denotes the expectation over the distribution $p$. Since this expectation is intractable, we need to approximate the Bayes posterior.

Let $r(A, B)$, or $r$ for short, be a trial distribution. The following functional with respect to $r$ is called the free energy:

$$F(r) = \left\langle \log \tfrac{r(A,B)}{p(V|A,B)p(A)p(B)} \right\rangle_{r(A,B)} = \left\langle \log \tfrac{r(A,B)}{p(A,B|V)} \right\rangle_{r(A,B)} - \log p(V). \tag{4}$$

In the last equation, the first term is the Kullback-Leibler (KL) divergence from the trial distribution to the Bayes posterior, and the second term is a constant. Therefore, minimizing the free energy (4) amounts to finding a distribution closest to the Bayes posterior in the sense of the KL divergence. A general approach to Bayesian approximate inference is to find the minimizer of the free energy (4) with respect to $r$ in some restricted function space.

In the VB approximation, the independence between the entangled parameter matrices $A$ and $B$ is assumed:

$$r = r(A)r(B). \tag{5}$$

Under this constraint, an iterative algorithm for minimizing the free energy (4) was derived [3, 11]. Let $\widehat{r}$ be such a minimizer, and we define the MF solution by the mean of the target matrix $U$:

$$\widehat{U} = \left\langle BA^\top \right\rangle_{\widehat{r}(A,B)}. \tag{6}$$

In the context of PCA where $V$ is a data matrix, the solution is given as the subspace spanned by $\widehat{U}$.

The MF model has hyperparameters $(C_A, C_B)$ in the priors (2). By manually choosing them, we can control regularization and sparsity of the solution (e.g., the PCA dimensions). A popular way to set the hyperparameter in the Bayesian framework is again based on the minimization of the free energy (4):

$$(\widehat{C}_A, \widehat{C}_B) = \text{argmin}_{C_A, C_B} \left(\min_r F(r; C_A, C_B|V)\right).$$

We refer to this method as an empirical VB (EVB) method. When the noise variance $\sigma^2$ is unknown, it can also be estimated as

$$\widehat{\sigma}^2 = \text{argmin}_{\sigma^2} \left(\min_{r,C_A,C_B} F(r; C_A, C_B, \sigma^2|V)\right).$$

## 3 Simple Closed-Form Solutions of VBMF

Recently, the global analytic solution of VBMF has been derived [16]. However, it is given as a solution of a *quartic* equation (Corollary 1 in [16]), and it is not easy to use for further analysis due to its complicated expression. In this section, we derive much simpler forms, which will be used for analyzing VB-PCA in the next section.

### 3.1 VB Solution

Our new analytic-form solution only involves linear and quadratic equations, which is summarized in the following theorem (the proof is omitted due to the space limitation):

**Theorem 1** *Let*

$$V = \sum_{h=1}^{H} \gamma_h \boldsymbol{\omega}_{b_h} \boldsymbol{\omega}_{a_h}^{\top} \tag{7}$$

*be the singular value decomposition (SVD) of $V$ with its singular values $\{\gamma_h\}$ arranged in non-increasing order, and the associated right and left singular vectors $\{\boldsymbol{\omega}_{a_h}, \boldsymbol{\omega}_{b_h}\}$. Then, the VB solution can be written as a truncated shrinkage SVD as follows:*

$$\widehat{U}^{\mathrm{VB}} = \sum_{h=1}^{H} \widehat{\gamma}_h^{\mathrm{VB}} \boldsymbol{\omega}_{b_h} \boldsymbol{\omega}_{a_h}^{\top}, \qquad where \qquad \widehat{\gamma}_h^{\mathrm{VB}} = \begin{cases} \breve{\gamma}_h^{\mathrm{VB}} & if \, \gamma_h \geq \underline{\gamma}_h^{\mathrm{VB}}, \\ 0 & otherwise. \end{cases} \tag{8}$$

*Here, the truncation threshold and the shrinkage estimator are, respectively, given by*

$$\underline{\gamma}_h^{\mathrm{VB}} = \sigma \sqrt{\frac{(L+M)}{2} + \frac{\sigma^2}{2c_{a_h}^2 c_{b_h}^2} + \sqrt{\left(\frac{(L+M)}{2} + \frac{\sigma^2}{2c_{a_h}^2 c_{b_h}^2}\right)^2 - LM}}, \tag{9}$$

$$\breve{\gamma}_h^{\mathrm{VB}} = \gamma_h \left(1 - \frac{\sigma^2}{2\gamma_h^2}\left(M + L + \sqrt{(M-L)^2 + \frac{4\gamma_h^2}{c_{a_h}^2 c_{b_h}^2}}\right)\right). \tag{10}$$

We can also derive a simple closed-form expression of the VB posterior (omitted).

### 3.2 EVB Solution

Combining Theorem 1 with the global EVB solution (Corollary 2 in [16]), we have the following theorem (the proof is omitted):

**Theorem 2** *Let*

$$\alpha = \frac{L}{M}, \tag{11}$$

*and let $\underline{\kappa} = \underline{\kappa}(\alpha) \, (> 1)$ be the zero-cross point of the following decreasing function:*

$$\Xi\left(\kappa; \alpha\right) = \Phi\left(\sqrt{\alpha}\kappa\right) + \Phi\left(\frac{\kappa}{\sqrt{\alpha}}\right), \qquad where \qquad \Phi(x) = \frac{\log(x+1)}{x} - \frac{1}{2}. \tag{12}$$

*Then, the EVB solution can be written as a truncated shrinkage SVD as follows:*

$$\widehat{U}^{\mathrm{EVB}} = \sum_{h=1}^{H} \widehat{\gamma}_h^{\mathrm{EVB}} \boldsymbol{\omega}_{b_h} \boldsymbol{\omega}_{a_h}^{\top}, \qquad where \qquad \widehat{\gamma}_h^{\mathrm{EVB}} = \begin{cases} \breve{\gamma}_h^{\mathrm{EVB}} & if \, \gamma_h \geq \underline{\gamma}^{\mathrm{EVB}}, \\ 0 & otherwise. \end{cases} \tag{13}$$

*Here, the truncation threshold and the shrinkage estimator are, respectively, given by*

$$\underline{\gamma}^{\mathrm{EVB}} = \sigma \sqrt{M + L + \sqrt{LM}\left(\underline{\kappa} + \frac{1}{\underline{\kappa}}\right)}, \tag{14}$$

$$\breve{\gamma}_h^{\mathrm{EVB}} = \frac{\gamma_h}{2}\left(1 - \frac{(M+L)\sigma^2}{\gamma_h^2} + \sqrt{\left(1 - \frac{(M+L)\sigma^2}{\gamma_h^2}\right)^2 - \frac{4LM\sigma^4}{\gamma_h^4}}\right). \tag{15}$$

The EVB threshold (14) involves $\underline{\kappa}$, which needs to be numerically computed. We can easily prepare a table of the values for $0 < \alpha \leq 1$ beforehand, like the cumulative Gaussian probability used in statistical tests. Fig. 2 shows the EVB threshold (14) by a red solid curve labeled as 'EVB'.

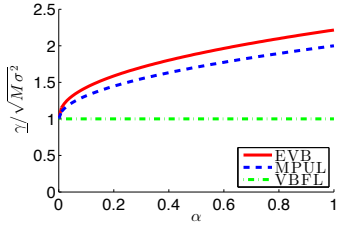
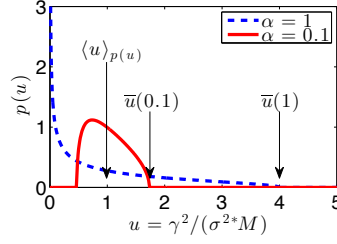
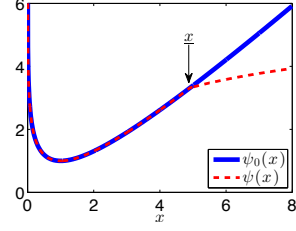

Figure 2: Thresholds.    Figure 3: Marčenko-Pastur law.    Figure 4: $\psi_0(x)$ and $\psi(x)$.

### 3.3 Large-Scale Limiting Behavior of EVB When Noise Variance Is Known

Here, we first introduce a result from random matrix theory [13, 22], and then discuss the behavior of EVB when the noise variance is *known*.

Assume that $\mathcal{E} \in \mathbb{R}^{L \times M}$ is a random matrix such that each element is independently drawn from a distribution with mean zero and variance $\sigma^{2*}$ (not necessarily Gaussian). Let $u_1, u_2, \ldots, u_L$ be the eigenvalues of $\frac{1}{M\sigma^{2*}}\mathcal{E}\mathcal{E}^\top$, and define the empirical distribution of the eigenvalues by

$$p(u) = \tfrac{1}{L}\left(\delta(u_1) + \delta(u_2) + \cdots + \delta(u_L)\right),$$

where $\delta(u)$ denotes the Dirac measure at $u$. Then the following proposition holds:

**Proposition 1 (Marčenko-Pastur law)** *[13, 22] In the large-scale limit when $L$ and $M$ go to infinity with its ratio $\alpha = L/M$ fixed, the probability measure of the empirical distribution of the eigenvalue $u$ of $\frac{1}{\sigma^{2*}M}\mathcal{E}\mathcal{E}^\top$ converges almost surely to*

$$p(u)du = \frac{\sqrt{(u-\underline{u})(\overline{u}-u)}}{2\pi\alpha u}\theta(\underline{u} < u < \overline{u})du, \tag{16}$$

*where $\underline{u} = (1 - \sqrt{\alpha})^2$, $\overline{u} = (1 + \sqrt{\alpha})^2$, and $\theta(\cdot)$ denotes an indicator function such that $\theta(condition) = 1$ if the condition is true and $\theta(condition) = 0$ otherwise.*

Fig. 3 shows the Marčenko-Pastur (MP) distributions for $\alpha = 0.1, 1$. The mean $\langle u \rangle_{p(u)} = 1$ (which is constant for any $0 < \alpha \le 1$) and the upper-limit $\overline{u} = \overline{u}(\alpha)$ for $\alpha = 0.1, 1$ are indicated by arrows. Note that the MP distribution appears for a *single* sample matrix; different from standard "large-sample" theories, we do not need many sample matrices (this property is sometimes called *self-averaging*). This single-sample property of the MP distribution is highly useful in our analysis because we are working with a single observation matrix in the MF scenario.

Proposition 1 states that all singular values of the random matrix $\mathcal{E}$ are almost surely upper-bounded by

$$\overline{\gamma}^{\mathrm{MPUL}} = \sqrt{M\sigma^{2*}\overline{u}} = (\sqrt{L} + \sqrt{M})\sigma^*, \tag{17}$$

which we call the Marčenko-Pastur upper-limit (MPUL). This property can be used for designing estimators robust against noise [10, 9]. Although EVB-PCA was proposed independently from the random matrix theory [3], its good performance can be proven with a related property to Proposition 1, as shown in Section 4.

When the noise variance is known, we can set the parameter to $\sigma = \sigma^*$ in Eq.(1). We depicted MPUL (17) for this case in Fig. 2. We can see that MPUL lower-bounds the EVB threshold (14) (actually this is true regardless of the value of $\underline{\kappa} > 0$). This implies a nice behavior of EVB, i.e., EVB eliminates all noise components in the large-scale limit. However, a simple optimal strategy—discarding the components with singular values smaller than $\overline{\gamma}^{\mathrm{MPUL}}$—outperforms EVB, because signals lying between the gap $[\overline{\gamma}^{\mathrm{MPUL}}, \underline{\gamma}^{\mathrm{EVB}})$ are discarded by EVB. Therefore, EVB is not very useful when $\sigma^{2*}$ is known. In Section 4, we investigate the behavior of EVB in a more practical and challenging situation where $\sigma^{2*}$ is unknown and is also estimated from observation.

In Fig. 2, we also depicted the VB threshold (9) with almost flat prior $c_{a_h}, c_{b_h} \to \infty$ (labeled as 'VBFL') for comparison. Actually, this coincides with the mean of the MP distribution, i.e., $\lim_{c_{a_h}, c_{a_h} \to \infty}(\underline{\gamma}_h^{\mathrm{VB}})^2/(M\sigma^2) = \langle u \rangle_{p(u)} = 1$. This implies that VBFL retains a lot of noise components, and does not perform well even when $\sigma^{2*}$ is known.

# 4 Analysis of EVB When Noise Variance Is Unknown

In this section, we derive bounds of the VB-based noise variance estimator, and obtain a sufficient condition for perfect dimensionality recovery in the large-scale limit.

## 4.1 Bounds of Noise Variance Estimator

The simple closed-form solution obtained in Section 3 is the global minimizer of the free energy (4), given $\sigma^2$. Using the solution, we can explicitly describe the free energy as a function of $\sigma^2$. We obtain the following theorem (the proof is omitted):

**Theorem 3** *The noise variance estimator $\widehat{\sigma}^{2\,\mathrm{EVB}}$ is the global minimizer of*

$$\Omega(\sigma^{-2}) = \sum_{h=1}^{H} \psi\left(\frac{\gamma_h^2}{M\sigma^2}\right) + \sum_{h=H+1}^{L} \psi_0\left(\frac{\gamma_h^2}{M\sigma^2}\right), \tag{18}$$

*where* $\quad \psi(x) = \psi_0(x) + \theta(x > \underline{x})\,\psi_1(x), \qquad \underline{x} = 1 + \alpha + \sqrt{\alpha}\left(\underline{\kappa} + \underline{\kappa}^{-1}\right), \tag{19}$

$$\psi_0(x) = x - \log x, \qquad \psi_1(x) = \log\left(\sqrt{\alpha}\kappa(x) + 1\right) + \alpha \log\left(\frac{\kappa(x)}{\sqrt{\alpha}} + 1\right) - \sqrt{\alpha}\kappa(x), \tag{20}$$

*$\underline{\kappa}$ is a constant defined in Theorem 2, and $\kappa(x)$ is a function of $x$ ($> \underline{x}$) defined by*

$$\kappa(x) = \frac{1}{2\sqrt{\alpha}}\left((x - (1+\alpha)) + \sqrt{(x - (1+\alpha))^2 - 4\alpha}\right). \tag{21}$$

Note that $\underline{x}$ and $\kappa(\gamma_h^2/(\sigma^2 M))$ are rescaled versions of the squared EVB threshold (14) and the EVB shrinkage estimator (15), respectively, i.e., $\underline{x} = (\underline{\gamma}^{\mathrm{EVB}})^2/(\sigma^2 M)$ and $\kappa(\gamma_h^2/(\sigma^2 M)) = \gamma_h \breve{\gamma}_h^{\mathrm{EVB}}/(\sigma^2 \sqrt{ML})$.

The functions $\psi_0(x)$ and $\psi(x)$ are depicted in Fig. 4. We can prove the convexity of $\psi_0(x)$ and quasi-convexity of $\psi(x)$, which are useful properties in our theoretical analysis.

Let $\widehat{H}^{\mathrm{EVB}}$ be the rank estimated by VB, which satisfies $\widehat{\gamma}_h^{\mathrm{EVB}} > 0$ for $h = 1, \ldots, \widehat{H}^{\mathrm{EVB}}$ and $\widehat{\gamma}_h^{\mathrm{EVB}} = 0$ for $h = \widehat{H}^{\mathrm{EVB}} + 1, \ldots, H$. Then we have the following theorem:

**Theorem 4** *$\widehat{H}^{\mathrm{EVB}}$ is upper-bounded as*

$$\widehat{H}^{\mathrm{EVB}} \leq \overline{H} = \min\left(\left\lceil\frac{L}{1+\alpha}\right\rceil - 1, H\right), \tag{22}$$

*and the noise variance estimator $\widehat{\sigma}^{2\,\mathrm{EVB}}$ is bounded as follows:*

$$\max\left(\underline{\sigma}_{\overline{H}+1}^2, \frac{\sum_{h=\overline{H}+1}^{L} \gamma_h^2}{M\left(L - \overline{H}(1+\alpha)\right)}\right) < \widehat{\sigma}^{2\,\mathrm{EVB}} \leq \frac{1}{LM}\sum_{h=1}^{L}\gamma_h^2, \tag{23}$$

$$\text{where} \quad \underline{\sigma}_h^2 = \begin{cases} \infty & \text{for } h = 0, \\ \frac{\gamma_h^2}{M\underline{x}} & \text{for } h = 1, \ldots, L, \\ 0 & \text{for } h = L + 1. \end{cases} \tag{24}$$

**(Sketch of proof)** First, we show that a global minimizer w.r.t. $\sigma^2$ exists in $(\gamma_L^2/M, \gamma_1^2/M)$, and it is a stationary point. Given a hypothetic $\widehat{H}$, the derivative of $\Omega$ w.r.t. $\sigma^{-2}$ is written as

$$\Theta(\sigma^{-2}) \equiv \frac{1}{L}\frac{\partial\Omega}{\partial\sigma^{-2}} = -\sigma^2 + \frac{\sum_{h=1}^{\widehat{H}}\gamma_h\left(\gamma_h - \breve{\gamma}_h^{\mathrm{EVB}}\right) + \sum_{h=\widehat{H}+1}^{L}\gamma_h^2}{LM}. \tag{25}$$

Eq.(15) implies the following bounds:

$$(M + L)\sigma^2 < \gamma_h\left(\gamma_h - \breve{\gamma}_h^{\mathrm{EVB}}\right) < (\sqrt{M} + \sqrt{L})^2\sigma^2 \quad \text{for } \gamma_h > \underline{\gamma}^{\mathrm{EVB}}, \tag{26}$$

which allows us to bound $\Theta$ by simple inequalities. Finding a condition prohibiting $\Theta$ to be zero proves the theorem. $\qquad\square$

Theorem 4 states that EVB discards the $(L - \lceil L/(1+\alpha)\rceil + 1) \geq 1$ smallest components, regardless of the observed values $\{\gamma_h\}$. For example, the half components are always discarded when the matrix is square (i.e., $\alpha = L/M = 1$). The smallest singular value $\gamma_L$ is always discarded, and $\widehat{\sigma}^{2\,\mathrm{EVB}} > \gamma_L^2/(M(L - (L-1)(1+\alpha)) > \gamma_L^2/M$ always holds.

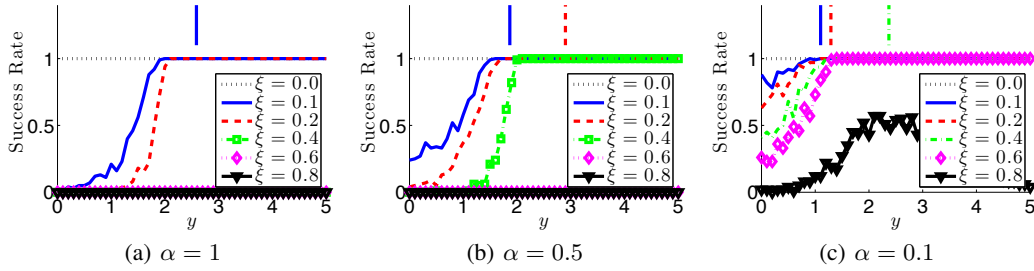

Figure 5: Success rate of dimensionality recovery in numerical simulation for $M = 200$. The threshold for the guaranteed recovery (the second inequality in Eq.(28)) is depicted with a vertical bar with the same color and line style.

## 4.2  Perfect Recovery Condition

Here, we derive a sufficient condition for perfect recovery of the true PCA dimensionality in the large-scale limit.

Assume that the observation matrix $V$ is generated as

$$V = U^* + \mathcal{E},  \qquad (27)$$

where $U^*$ is a *true* signal matrix with rank $H^*$ and the singular values $\{\gamma_h^*\}$, and each element of the noise matrix $\mathcal{E}$ is subject to a distribution with mean zero and variance $\sigma^{2*}$. We rely on a result [2, 5] on the eigenvalue distribution of the *spiked covariance model* [8]. The following theorem guarantees the accuracy of VB-PCA:

**Theorem 5** *Assume $H \geq H^*$ (i.e., we set the rank of the MF model sufficiently large), and denote the relevant rank (dimensionality) ratio by*

$$\xi = \frac{H^*}{L}.$$

*In the large-scale limit with finite $\alpha$ and $H^*$, EVB implemented with a local search algorithm for the noise variance $\sigma^2$ estimation almost surely recovers the true rank, i.e., $\widehat{H}^{\mathrm{EVB}} = H^*$, if $\xi = 0$ or*

$$\xi < \frac{1}{\underline{x}}  \quad and  \quad \gamma_{H^*}^{*2} > \left( \frac{x-1}{1-\underline{x}\xi} - \alpha \right) \cdot M\sigma^{2*},  \qquad (28)$$

*where $\underline{x}$ is defined in Eq.(19).*

**(Sketch of proof)** We first show that, in the large-scale limit and when $\xi = 0$, Eq.(25) is equal to zero if and only if $\sigma^2 = \sigma^{2*}$. This means the perfect recovery in the no-signal case. $\underline{\sigma}_h^2$ defined in Eq.(24) is actually the thresholding point of estimated $\widehat{\sigma}^2$ for the $h$-th component to be discarded. Therefore, $\widehat{H}^{\mathrm{EVB}} = H^*$ if and only if $\underline{\sigma}_{H^*+1}^2 < \widehat{\sigma}^2 < \underline{\sigma}_{H^*}^2$. Using Eq.(26), we can obtain a sufficient condition that a local minimum exists only in this range, which proves the theorem. □

Note that $\xi \to 0$ in the large scale limit. However, we treated $\xi$ as a positive value in Theorem 5, hoping that the obtained result can approximately hold in a practical situation when $L$ and $M$ are large but finite. The obtained result well explains the dependency on $\xi$ in the numerical simulation below.

Theorem 5 guarantees that, if the true rank $H^*$ is small enough compared with $L$ and the smallest signal $\gamma_{H^*}^*$ is large enough compared with $\sigma^{2*}$, VB-PCA works perfectly. It is important to note that, although the objective function (18) is non-convex and possibly multimodal in general, perfect recovery does not require global search, but only a local search, of the objective function for noise variance estimation.

Fig. 5 shows numerical results for $M = 200$ and $\alpha = 1, 0.5, 0.1$. $\mathcal{E}$ was drawn from the Gaussian distribution with variance $\sigma^{2*} = 1$, and signal singular values were drawn from the uniform distribution on $[yM\sigma^{2*}, 10M]$ for different $y$ (the horizontal axis of the graphs indicates $y$). The vertical axis indicates the success rate of dimensionality recovery, i.e., $\widehat{H}^{\mathrm{EVB}} = H^*$, over 100 trials. If the condition for $\xi$ (the first inequality in Eq.(28)) is violated, the corresponding line is depicted with markers. Otherwise, the threshold of $y$ for the guaranteed recovery (the second inequality in Eq.(28)) is indicated by a vertical bar with the same color and line style. We can see that the guarantee by Theorem 5 approximately holds even in this small matrix size, although it is slightly conservative.

# 5 Discussion

Here, we discuss implementation of VB-PCA, and the origin of sparsity of VB.

## 5.1 Implementation

Implementation of VB-PCA (VB-MF) based on the result given in [16] involves a quartic equation. This means that we need to use a highly complicated analytic-form solution, derived by, e.g., Ferrari's method, of a quartic equation, or rely on a numerical quartic solver, which is computationally less efficient. The theorems we gave in this paper can actually simplify the implementation greatly.

A table of $\underline{\kappa}$ defined in Theorem 2 should be prepared beforehand. Given an observed matrix $V$, we perform SVD and obtain the singular values. After that, in our new implementation, we first directly estimate the noise variance based on Theorem 3, using any 1-D local search algorithm—Theorem 4 helps restrict the search range. Then we obtain the noise variance estimator $\widehat{\sigma}^{2\,\mathrm{EVB}}$. For a dimensionality reduction purpose, simply discarding all the components such that $\underline{\sigma}_h^2 < \widehat{\sigma}^{2\,\mathrm{EVB}}$ gives the result (here $\underline{\sigma}_h^2$ is defined by Eq.(24)). When the estimator $\widehat{U}^{\mathrm{EVB}}$ is needed, Theorem 2 gives the result for $\sigma^2 = \widehat{\sigma}^{2\,\mathrm{EVB}}$. The VB posterior is also easily computed (its description is omitted). In this way, we can perform VB-PCA, whose performance is guaranteed, very easily.

## 5.2 Origin of *Exact* Sparsity

Sparsity is regarded as a practical advantage of VB. Nevertheless, as discussed in Section 1, it is not necessarily a property inherent in the rigorous Bayesian estimation. Actually, at least in MF, the sparsity is induced by the independence assumption between $A$ and $B$. Let us go back to Fig.1, where the Bayes posterior for $V = 1$ is shown in the left graph. In this scalar factorization model, the probability mass in the first and the third quadrants pulls the estimator $\widehat{U} = BA$ toward the positive direction, and the mass in the second and the fourth quadrants toward the negative direction. Since the Bayes posterior is skewed and more mass is put in the first and the third quadrants, it is natural that the Bayesian estimator $\widehat{\gamma} = \langle BA \rangle_{p(A,B|V)}$ is positive. This is true even if $V > 0$ is very small. On the other hand, the VB posterior (the middle graph) is prohibited to be skewed because of the independent assumption, and thus it has to wait until $V$ grows so that one of the modes has a enough probability mass. This is the cause of sparsity in VBMF. The Bayes posterior (left graph) implies that, if we restrict the posterior to be Gaussian, but allow to have correlation between $A$ and $B$, exact sparsity will not be observed.

It is observed that the Bayesian estimation gives a sparse solution when the hyper parameters $(C_A, C_B)$ are optimized. This estimator is also depicted as blue diamonds labeled as EFB (empirical fully-Bayesian) in the right graph of Fig.1. Note that, in this case, the independence between $A$ and $C_A^{-1/2}$ (as well as $B$ and $C_B^{-1/2}$), which are strongly correlated in the prior (2) and hence in the Bayes posterior, is forced—the point estimation of $C_A$ (as well as $C_B$) breaks the correlation because approximating by the delta function induces the independence from all other parameters. Further investigation on the relation between the independence constraint and the sparsity induction is our future work.

# 6 Conclusion

In this paper, we considered the variational Bayesian PCA (VB-PCA) when the noise variance is unknown. Analyzing the behavior of the noise variance estimator, we derived a sufficient condition for VB-PCA to perfectly recover the true dimensionality. Our result theoretically supports the usefulness of VB-PCA. In our theoretical analysis, we obtained bounds for a noise variance estimator and simple closed-form solutions for other parameters, which were shown to be useful for better implementation of VB-PCA.

## Acknowledgments

SN, RT, and MS thank the support from MEXT Kakenhi 23120004, MEXT Kakenhi 22700138, and the FIRST program, respectively. SDB was supported by a Beckman Postdoctoral Fellowship.

## Footnotes

[1]Also in mixture models, *inappropriate* model pruning by VB approximation was discussed [12].

[2]If the noise variance is known, we can actually show that dimensionality selection by VB-PCA is outperformed by a naive strategy (see Section 3.3). This means that VB-PCA is not very useful in this setting.

[3]When the number of samples is larger (smaller) than the data dimensionality in the PCA setting, the observation matrix $V$ should consist of the columns (rows), each of which corresponds to each sample.

# References

[1] H. Attias. Inferring parameters and structure of latent variable models by variational Bayes. In *Proceedings of the Fifteenth Conference Annual Conference on Uncertainty in Artificial Intelligence (UAI-99)*, pages 21–30, San Francisco, CA, 1999. Morgan Kaufmann.

[2] J. Baik and J. W. Silverstein. Eigenvalues of large sample covariance matrices of spiked population models. *Journal of Multivariate Analysis*, 97(6):1382–1408, 2006.

[3] C. M. Bishop. Variational principal components. In *Proc. of ICANN*, volume 1, pages 514–509, 1999.

[4] Z. Ghahramani and M. J. Beal. Graphical models and variational methods. In *Advanced Mean Field Methods*, pages 161–177. MIT Press, 2001.

[5] D. C. Hoyle. Automatic PCA dimension selection for high dimensional data and small sample sizes. *Journal of Machine Learning Research*, 9:2733–2759, 2008.

[6] A. Ilin and T. Raiko. Practical approaches to principal component analysis in the presence of missing values. *JMLR*, 11:1957–2000, 2010.

[7] T. S. Jaakkola and M. I. Jordan. Bayesian parameter estimation via variational methods. *Statistics and Computing*, 10:25–37, 2000.

[8] I. M. Johnstone. On the distribution of the largest eigenvalue in principal components analysis. *Annals of Statistics*, 29:295–327, 2001.

[9] N. El Karoui. Spectrum estimation for large dimensional covariance matrices using random matrix theory. *Annals of Statistics*, 36(6):2757–2790, 2008.

[10] O. Ledoit and M. Wolf. A well-conditioned estimator for large-dimensional covariance matrices. *Journal of Multivariate Analysis*, 88(2):365–411, 2004.

[11] Y. J. Lim and T. W. Teh. Variational Bayesian approach to movie rating prediction. In *Proceedings of KDD Cup and Workshop*, 2007.

[12] D. J. C. Mackay. Local minima, symmetry-breaking, and model pruning in variational free energy minimization. Available from `http://www.inference.phy.cam.ac.uk/mackay/minima.pdf`. 2001.

[13] V. A. Marcenko and L. A. Pastur. Distribution of eigenvalues for some sets of random matrices. *Mathematics of the USSR-Sbornik*, 1(4):457–483, 1967.

[14] T. P. Minka. Automatic choice of dimensionality for PCA. In *Advances in NIPS*, volume 13, pages 598–604. MIT Press, 2001.

[15] S. Nakajima and M. Sugiyama. Theoretical analysis of Bayesian matrix factorization. *Journal of Machine Learning Research*, 12:2579–2644, 2011.

[16] S. Nakajima, M. Sugiyama, and S. D. Babacan. Global solution of fully-observed variational Bayesian matrix factorization is column-wise independent. In *Advances in Neural Information Processing Systems 24*, 2011.

[17] S. Nakajima, M. Sugiyama, and S. D. Babacan. On Bayesian PCA: Automatic dimensionality selection and analytic solution. In *Proceedings of 28th International Conference on Machine Learning (ICML2011)*, Bellevue, WA, USA, Jun. 28–Jul.2 2011.

[18] S. Roweis and Z. Ghahramani. A unifying review of linear Gaussian models. *Neural Computation*, 11:305–345, 1999.

[19] R. Salakhutdinov and A. Mnih. Probabilistic matrix factorization. In J. C. Platt, D. Koller, Y. Singer, and S. Roweis, editors, *Advances in Neural Information Processing Systems 20*, pages 1257–1264, Cambridge, MA, 2008. MIT Press.

[20] M. Sato, T. Yoshioka, S. Kajihara, K. Toyama, N. Goda, K. Doya, and M. Kawato. Hierarchical Bayesian estimation for MEG inverse problem. *Neuro Image*, 23:806–826, 2004.

[21] M. E. Tipping and C. M. Bishop. Probabilistic principal component analysis. *Journal of the Royal Statistical Society*, 61:611–622, 1999.

[22] K. W. Wachter. The strong limits of random matrix spectra for sample matrices of independent elements. *Annals of Probability*, 6:1–18, 1978.

